# Learning Non-Linear Combinations of Kernels

**Corinna Cortes**
Google Research
76 Ninth Ave
New York, NY 10011
corinna@google.com

**Mehryar Mohri**
Courant Institute and Google
251 Mercer Street
New York, NY 10012
mohri@cims.nyu.edu

**Afshin Rostamizadeh**
Courant Institute and Google
251 Mercer Street
New York, NY 10012
rostami@cs.nyu.edu

## Abstract

This paper studies the general problem of learning kernels based on a polynomial combination of base kernels. We analyze this problem in the case of regression and the kernel ridge regression algorithm. We examine the corresponding learning kernel optimization problem, show how that minimax problem can be reduced to a simpler minimization problem, and prove that the global solution of this problem always lies on the boundary. We give a projection-based gradient descent algorithm for solving the optimization problem, shown empirically to converge in few iterations. Finally, we report the results of extensive experiments with this algorithm using several publicly available datasets demonstrating the effectiveness of our technique.

## 1 Introduction

Learning algorithms based on kernels have been used with much success in a variety of tasks [17,19]. Classification algorithms such as support vector machines (SVMs) [6, 10], regression algorithms, e.g., kernel ridge regression and support vector regression (SVR) [16, 22], and general dimensionality reduction algorithms such as kernel PCA (KPCA) [18] all benefit from kernel methods. Positive definite symmetric (PDS) kernel functions implicitly specify an inner product in a high-dimension Hilbert space where large-margin solutions are sought. So long as the kernel function used is PDS, convergence of the training algorithm is guaranteed.

However, in the typical use of these kernel method algorithms, the choice of the PDS kernel, which is crucial to improved performance, is left to the user. A less demanding alternative is to require the user to instead specify a family of kernels and to use the training data to select the most suitable kernel out of that family. This is commonly referred to as the problem of *learning kernels*.

There is a large recent body of literature addressing various aspects of this problem, including deriving efficient solutions to the optimization problems it generates and providing a better theoretical analysis of the problem both in classification and regression [1, 8, 9, 11, 13, 15, 21]. With the exception of a few publications considering infinite-dimensional kernel families such as hyperkernels [14] or general convex classes of kernels [2], the great majority of analyses and algorithmic results focus on learning finite *linear* combinations of base kernels as originally considered by [12]. However, despite the substantial progress made in the theoretical understanding and the design of efficient algorithms for the problem of learning such linear combinations of kernels, no method seems to reliably give improvements over baseline methods. For example, the learned linear combination does not consistently outperform either the uniform combination of base kernels or simply the best single base kernel (see, for example, UCI dataset experiments in [9, 12], see also NIPS 2008 workshop). This suggests exploring other *non-linear* families of kernels to obtain consistent and significant performance improvements.

Non-linear combinations of kernels have been recently considered by [23]. However, here too, experimental results have not demonstrated a consistent performance improvement for the general

learning task. Another method, hierarchical multiple learning [3], considers learning a linear combination of an exponential number of linear kernels, which can be efficiently represented as a product of sums. Thus, this method can also be classified as learning a non-linear combination of kernels. However, in [3] the base kernels are restricted to *concatenation* kernels, where the base kernels apply to disjoint subspaces. For this approach the authors provide an effective and efficient algorithm and some performance improvement is actually observed for regression problems in very high dimensions.

This paper studies the general problem of learning kernels based on a polynomial combination of base kernels. We analyze that problem in the case of regression using the kernel ridge regression (KRR) algorithm. We show how to simplify its optimization problem from a minimax problem to a simpler minimization problem and prove that the global solution of the optimization problem always lies on the boundary. We give a projection-based gradient descent algorithm for solving this minimization problem that is shown empirically to converge in few iterations. Furthermore, we give a necessary and sufficient condition for this algorithm to reach a global optimum. Finally, we report the results of extensive experiments with this algorithm using several publicly available datasets demonstrating the effectiveness of our technique.

The paper is structured as follows. In Section 2, we introduce the non-linear family of kernels considered. Section 3 discusses the learning problem, formulates the optimization problem, and presents our solution. In Section 4, we study the performance of our algorithm for learning non-linear combinations of kernels in regression (NKRR) on several publicly available datasets.

## 2  Kernel Family

This section introduces and discusses the family of kernels we consider for our learning kernel problem. Let $K_1, \ldots, K_p$ be a finite set of kernels that we combine to define more complex kernels. We refer to these kernels as *base kernels*. In much of the previous work on learning kernels, the family of kernels considered is that of linear or convex combinations of some base kernels. Here, we consider polynomial combinations of higher degree $d \geq 1$ of the base kernels with non-negative coefficients of the form:

$$K_{\boldsymbol{\mu}} = \sum_{0 \leq k_1 + \cdots + k_p \leq d,\, k_i \geq 0,\, i \in [0,p]} \mu_{k_1 \cdots k_p} K_1^{k_1} \cdots K_p^{k_p}, \qquad \mu_{k_1 \cdots k_p} \geq 0. \tag{1}$$

Any kernel function $K_{\boldsymbol{\mu}}$ of this form is PDS since products and sums of PDS kernels are PDS [4]. Note that $K_{\boldsymbol{\mu}}$ is in fact a linear combination of the PDS kernels $K_1^{k_1} \cdots K_p^{k_p}$. However, the number of coefficients $\mu_{k_1 \cdots k_p}$ is in $O(p^d)$, which may be too large for a reliable estimation from a sample of size $m$. Instead, we can assume that for some subset $I$ of all $p$-tuples $(k_1, \ldots, k_p)$, $\mu_{k_1 \cdots k_p}$ can be written as a product of non-negative coefficients $\mu_1, \ldots, \mu_p$: $\mu_{k_1 \cdots k_p} = \mu_1^{k_1} \cdots \mu_p^{k_p}$. Then, the general form of the polynomial combinations we consider becomes

$$K = \sum_{(k_1, \ldots, k_p) \in I} \mu_1^{k_1} \cdots \mu_p^{k_p} K_1^{k_1} \cdots K_p^{k_p} + \sum_{(k_1, \ldots, k_p) \in J} \mu_{k_1 \cdots k_p} K_1^{k_1} \cdots K_p^{k_p}, \tag{2}$$

where $J$ denotes the complement of the subset $I$. The total number of free parameters is then reduced to $p + |J|$. The choice of the set $I$ and its size depends on the sample size $m$ and possible prior knowledge about relevant kernel combinations. The second sum of equation (2) defining our general family of kernels represents a linear combination of PDS kernels. In the following, we focus on kernels that have the form of the first sum and that are thus non-linear in the parameters $\mu_1, \ldots, \mu_p$. More specifically, we consider kernels $K_{\boldsymbol{\mu}}$ defined by

$$K_{\boldsymbol{\mu}} = \sum_{k_1 + \cdots + k_p = d} \mu_1^{k_1} \cdots \mu_p^{k_p} K_1^{k_1} \cdots K_p^{k_p}, \tag{3}$$

where $\boldsymbol{\mu} = (\mu_1, \ldots, \mu_p)^\top \in \mathbb{R}^p$. For the ease of presentation, our analysis is given for the case $d = 2$, where the quadratic kernel can be given the following simpler expression:

$$K_{\boldsymbol{\mu}} = \sum_{k,l=1}^{p} \mu_k \mu_l \, K_k K_l. \tag{4}$$

But, the extension to higher-degree polynomials is straightforward and our experiments include results for degrees $d$ up to $4$.

# 3 Algorithm for Learning Non-Linear Kernel Combinations

## 3.1 Optimization Problem

We consider a standard regression problem where the learner receives a training sample of size $m$, $S = ((x_1, y_1), \ldots, (x_m, y_m)) \in (X \times Y)^m$, where $X$ is the input space and $Y \in \mathbb{R}$ the label space. The family of hypotheses $H_{\boldsymbol{\mu}}$ out of which the learner selects a hypothesis is the reproducing kernel Hilbert space (RKHS) associated to a PDS kernel function $K_{\boldsymbol{\mu}} \colon X \times X \to \mathbb{R}$ as defined in the previous section. Unlike standard kernel-based regression algorithms however, here, both the parameter vector $\boldsymbol{\mu}$ defining the kernel $K_{\boldsymbol{\mu}}$ and the hypothesis are learned using the training sample $S$.

The learning kernel algorithm we consider is derived from kernel ridge regression (KRR). Let $\mathbf{y} = [y_1, \ldots, y_m]^\top \in \mathbb{R}^m$ denote the vector of training labels and let $\mathbf{K}_{\boldsymbol{\mu}}$ denote the Gram matrix of the kernel $K_{\boldsymbol{\mu}}$ for the sample $S$: $[\mathbf{K}_{\boldsymbol{\mu}}]_{i,j} = K_{\boldsymbol{\mu}}(x_i, x_j)$, for all $i, j \in [1, m]$. The standard KRR dual optimization algorithm for a fixed kernel matrix $\mathbf{K}_{\boldsymbol{\mu}}$ is given in terms of the Lagrange multipliers $\boldsymbol{\alpha} \in \mathbb{R}^m$ by [16]:

$$\max_{\boldsymbol{\alpha} \in \mathbb{R}^m} -\boldsymbol{\alpha}^\top (\mathbf{K}_{\boldsymbol{\mu}} + \lambda \mathbf{I}) \boldsymbol{\alpha} + 2\boldsymbol{\alpha}^\top \mathbf{y} \tag{5}$$

The related problem of learning the kernel $\mathbf{K}_{\boldsymbol{\mu}}$ concomitantly can be formulated as the following min-max optimization problem [9]:

$$\min_{\boldsymbol{\mu} \in \mathcal{M}} \max_{\boldsymbol{\alpha} \in \mathbb{R}^m} -\boldsymbol{\alpha}^\top (\mathbf{K}_{\boldsymbol{\mu}} + \lambda \mathbf{I}) \boldsymbol{\alpha} + 2\boldsymbol{\alpha}^\top \mathbf{y}, \tag{6}$$

where $\mathcal{M}$ is a positive, bounded, and convex set. The positivity of $\boldsymbol{\mu}$ ensures that $\mathbf{K}_{\boldsymbol{\mu}}$ is positive semi-definite (PSD) and its boundedness forms a regularization controlling the norm of $\boldsymbol{\mu}$.[1] Two natural choices for the set $\mathcal{M}$ are the norm-1 and norm-2 bounded sets,

$$\mathcal{M}_1 = \{\boldsymbol{\mu} \mid \boldsymbol{\mu} \succeq 0 \ \wedge \ \|\boldsymbol{\mu} - \boldsymbol{\mu}_0\|_1 \leq \Lambda\} \tag{7}$$

$$\mathcal{M}_2 = \{\boldsymbol{\mu} \mid \boldsymbol{\mu} \succeq 0 \ \wedge \ \|\boldsymbol{\mu} - \boldsymbol{\mu}_0\|_2 \leq \Lambda\}. \tag{8}$$

These definitions include an offset parameter $\boldsymbol{\mu}_0$ for the weights $\boldsymbol{\mu}$. Some natural choices for $\boldsymbol{\mu}_0$ are: $\boldsymbol{\mu}_0 = \mathbf{0}$, or $\boldsymbol{\mu}_0 / \|\boldsymbol{\mu}_0\| = \mathbf{1}$. Note that here, since the objective function is not linear in $\boldsymbol{\mu}$, the norm-1-type regularization may not lead to a sparse solution.

## 3.2 Algorithm Formulation

For learning linear combinations of kernels, a typical technique consists of applying the minimax theorem to permute the $\min$ and $\max$ operators, which can lead to optimization problems computationally more efficient to solve [8, 12]. However, in the non-linear case we are studying, this technique is unfortunately not applicable.

Instead, our method for learning non-linear kernels and solving the min-max problem in equation (6) consists of first directly solving the inner maximization problem. In the case of KRR for any fixed $\boldsymbol{\mu}$ the optimum is given by

$$\boldsymbol{\alpha} = (\mathbf{K}_{\boldsymbol{\mu}} + \lambda \mathbf{I})^{-1} \mathbf{y}. \tag{9}$$

Plugging the optimal expression of $\boldsymbol{\alpha}$ in the min-max optimization yields the following equivalent minimization in terms of $\boldsymbol{\mu}$ only:

$$\min_{\boldsymbol{\mu} \in \mathcal{M}} \quad F(\boldsymbol{\mu}) = \mathbf{y}^\top (\mathbf{K}_{\boldsymbol{\mu}} + \lambda \mathbf{I})^{-1} \mathbf{y}. \tag{10}$$

We refer to this optimization as the NKRR problem. Although the original min-max problem has been reduced to a simpler minimization problem, the function $F$ is not convex in general as illustrated by Figure 1. For small values of $\boldsymbol{\mu}$, concave regions are observed. Thus, standard interior-point or gradient methods are not guaranteed to be successful at finding a global optimum.

In the following, we give an analysis which shows that under certain conditions it is however possible to guarantee the convergence of a gradient-descent type algorithm to a global minimum.

Algorithm 1 illustrates a general gradient descent algorithm for the norm-2 bounded setting which projects $\boldsymbol{\mu}$ back to the feasible set $\mathcal{M}_2$ after each gradient step (projecting to $\mathcal{M}_1$ is very similar).

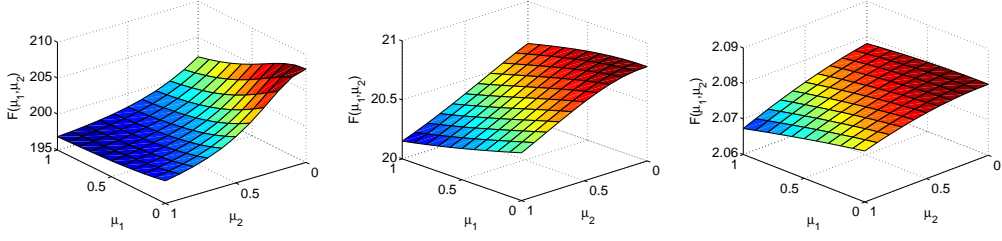

Figure 1: Example plots for $F$ defined over two linear base kernels generated from the first two features of the sonar dataset. From left to right $\lambda = 1, 10, 100$. For larger values of $\lambda$ it is clear that there are in fact concave regions of the function near $\mathbf{0}$.

---
**Algorithm 1** Projection-based Gradient Descent Algorithm
---

  **Input:** $\boldsymbol{\mu}_{\text{init}} \in \mathcal{M}_2, \eta \in [0,1], \epsilon > 0, \mathbf{K}_k, k \in [1,p]$
  $\boldsymbol{\mu}' \leftarrow \boldsymbol{\mu}_{\text{init}}$
  **repeat**
    $\boldsymbol{\mu} \leftarrow \boldsymbol{\mu}'$
    $\boldsymbol{\mu}' \leftarrow -\eta\nabla F(\boldsymbol{\mu}) + \boldsymbol{\mu}$
    $\forall k, \mu'_k \leftarrow \max(0, \mu'_k)$
    normalize $\boldsymbol{\mu}'$, s.t. $\|\boldsymbol{\mu}' - \boldsymbol{\mu}_0\| = \Lambda$
  **until** $\|\boldsymbol{\mu}' - \boldsymbol{\mu}\| < \epsilon$

---

In Algorithm 1 we have fixed the step size $\eta$, however this can be adjusted at each iteration via a line-search. Furthermore, as shown later, the thresholding step that forces $\boldsymbol{\mu}'$ to be positive is unnecessary since $\nabla F$ is never positive.

Note that Algorithm 1 is simpler than the wrapper method proposed by [20]. Because of the closed form expression (10), we do not alternate between solving for the dual variables and performing a gradient step in the kernel parameters. We only need to optimize with respect to the kernel parameters.

### 3.3 Algorithm Properties

We first explicitly calculate the gradient of the objective function for the optimization problem (10). In what follows, $\circ$ denotes the Hadamard (pointwise) product between matrices.

**Proposition 1.** *For any $k \in [1,p]$, the partial derivative of $F\colon \boldsymbol{\mu} \to \mathbf{y}^\top(\mathbf{K}_{\boldsymbol{\mu}} + \lambda\mathbf{I})^{-1}\mathbf{y}$ with respect to $\mu_i$ is given by*

$$\frac{\partial F}{\partial \mu_k} = -2\boldsymbol{\alpha}^\top \mathbf{U}_k \boldsymbol{\alpha}, \tag{11}$$

*where $\mathbf{U}_k = \left(\sum_{r=1}^{p}(\mu_r\mathbf{K}_r) \circ \mathbf{K}_k\right)$.*

*Proof.* In view of the identity $\nabla_{\mathbf{M}}\operatorname{Tr}(\mathbf{y}^\top\mathbf{M}^{-1}\mathbf{y}) = -\mathbf{M}^{-1^\top}\mathbf{y}\mathbf{y}^\top\mathbf{M}^{-1^\top}$, we can write:

$$
\begin{aligned}
\frac{\partial F}{\partial \mu_k} &= \operatorname{Tr}\left[\frac{\partial \mathbf{y}^\top(\mathbf{K}_{\boldsymbol{\mu}} + \lambda\mathbf{I})^{-1}\mathbf{y}}{\partial(\mathbf{K}_{\boldsymbol{\mu}} + \lambda\mathbf{I})} \frac{\partial(\mathbf{K}_{\boldsymbol{\mu}} + \lambda\mathbf{I})}{\partial \mu_k}\right]\\
&= -\operatorname{Tr}\left[(\mathbf{K}_{\boldsymbol{\mu}} + \lambda\mathbf{I})^{-1}\mathbf{y}\mathbf{y}^\top(\mathbf{K}_{\boldsymbol{\mu}} + \lambda\mathbf{I})^{-1}\frac{\partial(\mathbf{K}_{\boldsymbol{\mu}} + \lambda\mathbf{I})}{\partial \mu_k}\right]\\
&= -\operatorname{Tr}\left[(\mathbf{K}_{\boldsymbol{\mu}} + \lambda\mathbf{I})^{-1}\mathbf{y}\mathbf{y}^\top(\mathbf{K}_{\boldsymbol{\mu}} + \lambda\mathbf{I})^{-1}\left(2\sum_{r=1}^{p}(\mu_r\mathbf{K}_r) \circ \mathbf{K}_k\right)\right]\\
&= -2\mathbf{y}^\top(\mathbf{K}_{\boldsymbol{\mu}} + \lambda\mathbf{I})^{-1}\left(\sum_{r=1}^{p}(\mu_r\mathbf{K}_r) \circ \mathbf{K}_k\right)(\mathbf{K}_{\boldsymbol{\mu}} + \lambda\mathbf{I})^{-1}\mathbf{y} = -2\boldsymbol{\alpha}^\top \mathbf{U}_k\boldsymbol{\alpha}. \qquad \square
\end{aligned}
$$

Matrix $\mathbf{U}_k$ just defined in proposition 1 is always PSD, thus $\frac{\partial F}{\partial \mu_k} \leq 0$ for all $i \in [1, p]$ and $\nabla F \leq 0$. As already mentioned, this fact obliterates the thresholding step in Algorithm 1. We now provide guarantees for convergence to a global optimum. We shall assume that $\lambda$ is strictly positive: $\lambda > 0$.

**Proposition 2.** *Any stationary point $\boldsymbol{\mu}^\star$ of the function $F\colon \boldsymbol{\mu} \to \mathbf{y}^\top (\mathbf{K}_{\boldsymbol{\mu}} + \lambda \mathbf{I})^{-1} \mathbf{y}$ necessarily maximizes $F$:*

$$F(\boldsymbol{\mu}^\star) = \max_{\boldsymbol{\mu}} F(\boldsymbol{\mu}) = \frac{\|\mathbf{y}\|^2}{\lambda}. \tag{12}$$

*Proof.* In view of the expression of the gradient given by Proposition 1, at any point $\boldsymbol{\mu}^\star$,

$$\boldsymbol{\mu}^{\star \top} \nabla F(\boldsymbol{\mu}^\star) = \boldsymbol{\alpha}^\top \sum_{i=1}^p \mu_k^\star \mathbf{U}_k \boldsymbol{\alpha} = \boldsymbol{\alpha}^\top \mathbf{K}_{\boldsymbol{\mu}^\star} \boldsymbol{\alpha}. \tag{13}$$

By definition, if $\boldsymbol{\mu}^\star$ is a stationary point, $\nabla F(\boldsymbol{\mu}^\star) = 0$, which implies $\boldsymbol{\mu}^{\star \top} \nabla F(\boldsymbol{\mu}^\star) = 0$. Thus, $\boldsymbol{\alpha}^\top \mathbf{K}_{\boldsymbol{\mu}^\star} \boldsymbol{\alpha} = 0$, which implies $\mathbf{K}_{\boldsymbol{\mu}^\star} \boldsymbol{\alpha} = 0$, that is

$$\mathbf{K}_{\boldsymbol{\mu}^\star} (\mathbf{K}_{\boldsymbol{\mu}^\star} + \lambda \mathbf{I})^{-1} \mathbf{y} = 0 \Leftrightarrow (\mathbf{K}_{\boldsymbol{\mu}^\star} + \lambda \mathbf{I} - \lambda \mathbf{I})(\mathbf{K}_{\boldsymbol{\mu}^\star} + \lambda \mathbf{I})^{-1} \mathbf{y} = 0 \tag{14}$$

$$\Leftrightarrow \mathbf{y} - \lambda (\mathbf{K}_{\boldsymbol{\mu}^\star} + \lambda \mathbf{I})^{-1} \mathbf{y} = \mathbf{0} \tag{15}$$

$$\Leftrightarrow (\mathbf{K}_{\boldsymbol{\mu}^\star} + \lambda \mathbf{I})^{-1} \mathbf{y} = \frac{\mathbf{y}}{\lambda}. \tag{16}$$

Thus, for any such stationary point $\boldsymbol{\mu}^\star$, $F(\boldsymbol{\mu}^\star) = \mathbf{y}^\top (\mathbf{K}_{\boldsymbol{\mu}^\star} + \lambda \mathbf{I})^{-1} \mathbf{y} = \frac{\mathbf{y}^\top \mathbf{y}}{\lambda}$, which is clearly a maximum. $\qquad \square$

We next show that there cannot be an interior stationary point, and thus any local minimum strictly within the feasible set, unless the function is constant.

**Proposition 3.** *If any point $\boldsymbol{\mu}^\star > 0$ is a stationary point of $F\colon \boldsymbol{\mu} \to \mathbf{y}^\top (\mathbf{K}_{\boldsymbol{\mu}} + \lambda \mathbf{I})^{-1} \mathbf{y}$, then the function is necessarily constant.*

*Proof.* Assume that $\boldsymbol{\mu}^\star > 0$ is a stationary point, then, by Proposition 2, $F(\boldsymbol{\mu}^\star) = \mathbf{y}^\top (\mathbf{K}_{\boldsymbol{\mu}^\star} + \lambda \mathbf{I})^{-1} \mathbf{y} = \frac{\mathbf{y}^\top \mathbf{y}}{\lambda}$, which implies that $\mathbf{y}$ is an eigenvector of $(\mathbf{K}_{\boldsymbol{\mu}^\star} + \lambda \mathbf{I})^{-1}$ with eigenvalue $\lambda^{-1}$. Equivalently, $\mathbf{y}$ is an eigenvector of $\mathbf{K}_{\boldsymbol{\mu}^\star} + \lambda \mathbf{I}$ with eigenvalue $\lambda$, which is equivalent to $\mathbf{y} \in \text{null}(\mathbf{K}_{\boldsymbol{\mu}^\star})$. Thus,

$$\mathbf{y}^\top \mathbf{K}_{\boldsymbol{\mu}^\star} \mathbf{y} = \sum_{k,l=1}^p \mu_k \mu_l \underbrace{\sum_{r,s=1}^m y_r y_s K_k(x_r, x_s) K_l(x_r, x_s)}_{(*)} = 0. \tag{17}$$

Since the product of PDS functions is also PDS, (*) must be non-negative. Furthermore, since by assumption $\mu_i > 0$ for all $i \in [1, p]$, it must be the case that the term (*) is equal to zero. Thus, equation 17 is equal to zero for all $\boldsymbol{\mu}$ and the function $F$ is equal to the constant $\|\mathbf{y}\|^2 / \lambda$. $\qquad \square$

The previous propositions are sufficient to show that the gradient descent algorithm will not become stuck at a local minimum while searching the interior of a convex set $\mathcal{M}$ and, furthermore, they indicate that the optimum is found at the boundary.

The following proposition gives a necessary and sufficient condition for the convexity of $F$ on a convex region $C$. If the boundary region defined by $\|\boldsymbol{\mu} - \boldsymbol{\mu}_0\| = \Lambda$ is contained in this convex region, then Algorithm 1 is guaranteed to converge to a global optimum. Let $\mathbf{u} \in \mathbb{R}^p$ represent an arbitrary direction of $\boldsymbol{\mu}$ in $C$. We simplify the analysis of convexity in the following derivation by separating the terms that depend on $\mathbf{K}_{\boldsymbol{\mu}}$ and those depending on $\mathbf{K}_u$, which arise when showing the positive semi-definiteness of the Hessian, i.e. $\mathbf{u}^\top \nabla^2 F \mathbf{u} \succeq 0$. We denote by $\otimes$ the Kronecker product of two matrices.

**Proposition 4.** *The function $F\colon \boldsymbol{\mu} \to \mathbf{y}^\top (\mathbf{K}_{\boldsymbol{\mu}} + \lambda \mathbf{I})^{-1} \mathbf{y}$ is convex over the convex set $C$ iff the following condition holds for all $\boldsymbol{\mu} \in C$ and all $\mathbf{u}$:*

$$\langle \mathbf{M}, \mathbf{N} - \widetilde{\mathbf{1}} \rangle_F \geq 0, \tag{18}$$

| Data | $m$ | $p$ | lin. base | lin. $\ell_1$ | lin. $\ell_2$ | quad. base | quad. $\ell_1$ | quad. $\ell_2$ |
|---|---|---|---|---|---|---|---|---|
| Parkinsons | 194 | 21 | $.70 \pm .03$ | $.70 \pm .04$ | $.70 \pm .03$ | $.65 \pm .03$ | $.66 \pm .03$ | $.64 \pm .03$ |
| Iono | 351 | 34 | $.82 \pm .03$ | $.81 \pm .04$ | $.81 \pm .03$ | $.62 \pm .05$ | $.62 \pm .05$ | $.60 \pm .05$ |
| Sonar | 208 | 60 | $.90 \pm .02$ | $.92 \pm .03$ | $.90 \pm .04$ | $.84 \pm .03$ | $.80 \pm .04$ | $.80 \pm .04$ |
| Breast | 683 | 9 | $.70 \pm .02$ | $.71 \pm .02$ | $.70 \pm .02$ | $.70 \pm .02$ | $.70 \pm .01$ | $.70 \pm .01$ |

Table 1: The square-root of the mean squared error is reported for each method and several datasets.

*where* $\mathbf{M} = \left(\mathbf{1} \otimes \mathrm{vec}(\boldsymbol{\alpha}\boldsymbol{\alpha}^\top)^\top\right) \circ (\mathbf{K_u} \otimes \mathbf{K_u})$, $\mathbf{N} = 4\left(\mathbf{1} \otimes \mathrm{vec}(\mathbf{V})^\top\right) \circ (\mathbf{K_\mu} \otimes \mathbf{K_\mu})$, *and* $\widetilde{\mathbf{1}}$ *is the matrix with zero-one entries constructed to select the terms* $[\mathbf{M}]_{ijkl}$ *where* $i = k$ *and* $j = l$, *i.e. it is non-zero only in the* $(i, j)$*th coordinate of the* $(i, j)$*th* $m \times m$ *block.*

*Proof.* For any $\mathbf{u} \in \mathbb{R}^p$ the expression of the Hessian of $F$ at the point $\boldsymbol{\mu} \in C$ can be derived from that of its gradient and shown to be

$$\mathbf{u}^\top(\nabla^2 F)\mathbf{u} = 4\boldsymbol{\alpha}^\top(\mathbf{K_\mu} \circ \mathbf{K_u})\mathbf{V}(\mathbf{K_\mu} \circ \mathbf{K_u})\boldsymbol{\alpha} - \boldsymbol{\alpha}^\top(\mathbf{K_u} \circ \mathbf{K_u})\boldsymbol{\alpha}. \tag{19}$$

Expanding each term, we obtain:

$$\boldsymbol{\alpha}^\top(\mathbf{K_\mu} \circ \mathbf{K_u})\mathbf{V}(\mathbf{K_\mu} \circ \mathbf{K_u})\boldsymbol{\alpha} = \sum_{i,j=1}^m \alpha_i \alpha_j \sum_{k,l=1}^m [\mathbf{K_\mu}]_{ik}[\mathbf{K_u}]_{ik}[\mathbf{V}]_{kl}[\mathbf{K_\mu}]_{ik}[\mathbf{K_\mu}]_{lj} \tag{20}$$

$$= \sum_{i,j,k,l=1}^m (\alpha_i \alpha_j [\mathbf{K_u}]_{ik}[\mathbf{K_u}]_{lj})([\mathbf{V}]_{kl}[\mathbf{K_\mu}]_{ik}[\mathbf{K_\mu}]_{lj}) \tag{21}$$

and $\boldsymbol{\alpha}^\top(\mathbf{K_u} \circ \mathbf{K_u})\boldsymbol{\alpha} = \sum_{i,j=1}^m \alpha_i \alpha_j [\mathbf{K_u}]_{ij}[\mathbf{K_u}]_{ij}$. Let $\mathbf{1} \in \mathbb{R}^{m^2}$ define the column vector of all ones and let $\mathrm{vec}(\mathbf{A})$ denote the vectorization of a matrix $\mathbf{A}$ by stacking its columns. Let the matrices $\mathbf{M}$ and $\mathbf{N}$ be defined as in the statement of the proposition. Then, $[\mathbf{M}]_{ijkl} = (\alpha_i \alpha_j [\mathbf{K_u}]_{ik}[\mathbf{K_u}]_{lj})$ and $[\mathbf{N}]_{ijkl} = [\mathbf{V}]_{kl}[\mathbf{K_\mu}]_{ik}[\mathbf{K_\mu}]_{lj}$. Then, in view of the definition of $\widetilde{\mathbf{1}}$, the terms of equation (19) can be represented with the Frobenius inner product,

$$\mathbf{u}^\top(\nabla^2 F)\mathbf{u} = \langle \mathbf{M}, \mathbf{N} \rangle_F - \langle \mathbf{M}, \widetilde{\mathbf{1}} \rangle_F = \langle \mathbf{M}, \mathbf{N} - \widetilde{\mathbf{1}} \rangle_F. \qquad \square$$

For any $\boldsymbol{\mu} \in \mathbb{R}^p$, let $\mathbf{K_\mu} = \sum_i \mu_i \mathbf{K}_i$ and let $\mathbf{V} = (\mathbf{K_\mu} + \lambda \mathbf{I})^{-1}$. We now show that the condition of Proposition 4 is satisfied for convex regions for which $\Lambda$, and therefore $\boldsymbol{\mu}$, is sufficiently large, in the case where $\mathbf{K_u}$ and $\mathbf{K_\mu}$ are diagonal. In that case, $\mathbf{M}$, $\mathbf{N}$ and $\mathbf{V}$ are diagonal as well and the condition of Proposition 4 can be rewritten as follows:

$$\sum_{i,j} [\mathbf{K_u}]_{ii}[\mathbf{K_u}]_{jj}\alpha_i \alpha_j (4[\mathbf{K_\mu}]_{ii}[\mathbf{K_\mu}]_{jj}\mathbf{V}_{ij} - \mathbf{1}_{i=j}) \geq 0. \tag{22}$$

Using the fact that $\mathbf{V}$ is diagonal, this inequality we can be further simplified

$$\sum_{i=1}^m [\mathbf{K_u}]_{ii}^2 \alpha_i^2 (4[\mathbf{K_\mu}]_{ii}^2 \mathbf{V}_{ii} - \mathbf{1}) \geq 0. \tag{23}$$

A sufficient condition for this inequality to hold is that each term $(4[\mathbf{K_\mu}]_{ii}^2 \mathbf{V}_{ii} - \mathbf{1})$ be non-negative, or equivalently that $4\mathbf{K_\mu}^2 \mathbf{V} - \mathbf{I} \succeq \mathbf{0}$, that is $\mathbf{K_\mu} \succeq \sqrt{\frac{\lambda}{3}}\mathbf{I}$. Therefore, it suffices to select $\boldsymbol{\mu}$ such that $\min_i \sum_{k=1}^p \mu_k [\mathbf{K}_k]_{ii} \geq \sqrt{\lambda/3}$.

## 4 Empirical Results

To test the advantage of learning non-linear kernel combinations, we carried out a number of experiments on publicly available datasets. The datasets are chosen to demonstrate the effectiveness of the algorithm under a number of conditions. For general performance improvement, we chose a number of UCI datasets frequently used in kernel learning experiments, e.g., [7, 12, 15]. For learning with thousands of kernels, we chose the sentiment analysis dataset of Blitzer et. al [5]. Finally, for learning with higher-order polynomials, we selected datasets with large number of examples such as *kin-8nm* from the Delve repository. The experiments were run on a 2.33 GHz Intel Xeon Processor with 2GB of RAM.

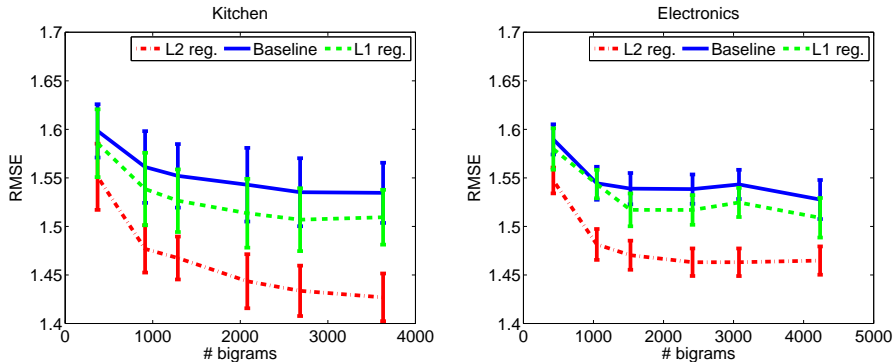

Figure 2: The performance of baseline and learned quadratic kernels (plus or minus one standard deviation) versus the number of bigrams (and kernels) used.

## 4.1 UCI Datasets

We first analyzed the performance of the kernels learned as quadratic combinations. For each dataset, features were scaled to lie in the interval $[0, 1]$. Then, both labels and features were centered. In the case of classification dataset, the labels were set to $\pm 1$ and the RMSE was reported. We associated a base kernel to each feature, which computes the product of this feature between different examples. We compared both linear and quadratic combinations, each with a baseline (uniform), norm-1-regularized and norm-2-regularized weighting using $\mu_0 = \mathbf{1}$ corresponding to the weights of the baseline kernel. The parameters $\lambda$ and $\Lambda$ were selected via 10-fold cross validation and the error reported was based on 30 random 50/50 splits of the entire dataset into training and test sets. For the gradient descent algorithm, we started with $\eta = 1$ and reduced it by a factor of $0.8$ if the step was found to be too large, i.e., the difference $\|\boldsymbol{\mu}' - \boldsymbol{\mu}\|$ increased. Convergence was typically obtained in less than 25 steps, each requiring a fraction of a second ($\sim 0.05$ seconds).

The results, which are presented in Table 1, are in line with previous ones reported for learning kernels on these datasets [7, 8, 12, 15]. They indicate that learning quadratic combination kernels can sometimes offer improvements and that it clearly does not degrade with respect to the performance of the baseline kernel. The learned quadratic combination performs well, particularly on tasks where the number of features was large compared to the number of points. This suggests that the learned kernel is better regularized than the plain quadratic kernel and can be advantageous is scenarios where over-fitting is an issue.

## 4.2 Text Based Dataset

We next analyzed a text-based task where features are frequent word $n$-grams. Each base kernel computes the product between the counts of a particular $n$-gram for the given pair of points. Such kernels have a direct connection to count-based rational kernels, as described in [8]. We used the sentiment analysis dataset of Blitzer et. al [5]. This dataset contains text-based user reviews found for products on `amazon.com`. Each text review is associated with a 0-5 star rating. The product reviews fall into two categories: electronics and kitchen-wares, each with 2,000 data-points. The data was not centered in this case since we wished to preserve the sparsity, which offers the advantage of significantly more efficient computations. A constant feature was included to act as an offset.

For each domain, the parameters $\lambda$ and $\Lambda$ were chosen via 10-fold cross validation on 1,000 points. Once these parameters were fixed, the performance of each algorithm was evaluated using 20 random 50/50 splits of the entire 2,000 points into training and test sets. We used the performance of the uniformly weighted quadratic combination kernel as a baseline, and showed the improvement when learning the kernel with norm-1 or norm-2 regularization using $\mu_0 = \mathbf{1}$ corresponding to the weights of the baseline kernel. As shown by Figure 2, the learned kernels significantly improved over the baseline quadratic kernel in both the kitchen and electronics categories. For this case too, the number of features was large in comparison with the number of points. Using 900 training points and about 3,600 bigrams, and thus kernels, each iteration of the algorithm took approximately 25

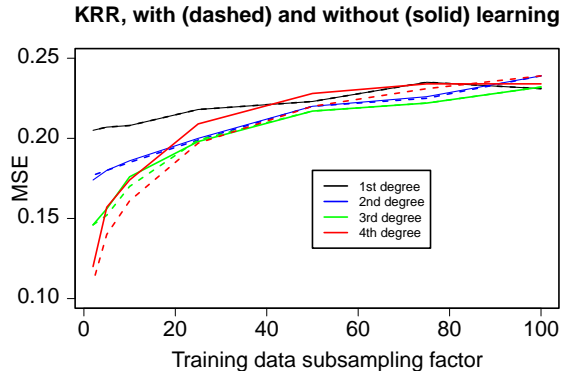

Figure 3: Performance on the *kin-8nm* dataset. For all polynomials, we compared un-weighted, standard KRR (solid lines) with norm-2 regularized kernel learning (dashed lines). For 4th degree polynomials we observed a clear performance improvement, especially for medium amount of training data (subsampling factor of 10-50). Standard deviations were typically in the order $0.005$, so the results were statistically significant.

seconds to compute with our Matlab implementation. When using norm-2 regularization, the algorithm generally converges in under 30 iterations, while the norm-1 regularization requires an even fewer number of iterations, typically less than 5.

### 4.3 Higher-order Polynomials

We finally investigated the performance of higher-order non-linear combinations. For these experiments, we used the *kin-8nm* dataset from the Delve repository. This dataset has 20,000 examples with 8 input features. Here too, we used polynomial kernels over the features, but this time we experimented with polynomials with degrees as high as 4. Again, we made the assumption that all coefficients of $\mu$ are in the form of products of $\mu_i$s (see Section 2), thus only 8 kernel parameters needed to be estimated.

We split the data into 10,000 examples for training and 10,000 examples for testing, and, to investigate the effect of the sample size on learning kernels, subsampled the training data so that only a fraction from 1 to 100 was used. The parameters $\lambda$ and $\Lambda$ were determined by 10-fold cross validation on the training data, and results are reported on the test data, see Figure 3. We used norm-2 regularization with $\mu_0 = 1$ and compare our results with those of uniformly weighted KRR.

For lower degree polynomials, the performance was essentially the same, but for 4th degree polynomials we observed a significant performance improvement of learning kernels over the uniformly weighted KRR, especially for a medium amount of training data (subsampling factor of 10-50). For the sake of readability, the standard deviations are not indicated in the plot. They were typically in the order of 0.005, so the results were statistically significant. This result corroborates the finding on the UCI dataset, that learning kernels is better regularized than plain unweighted KRR and can be advantageous is scenarios where overfitting is an issue.

## 5 Conclusion

We presented an analysis of the problem of learning polynomial combinations of kernels in regression. This extends learning kernel ideas and helps explore kernel combinations leading to better performance. We proved that the global solution of the optimization problem always lies on the boundary and gave a simple projection-based gradient descent algorithm shown empirically to converge in few iterations. We also gave a necessary and sufficient condition for that algorithm to converge to a global optimum. Finally, we reported the results of several experiments on publicly available datasets demonstrating the benefits of learning polynomial combinations of kernels. We are well aware that this constitutes only a preliminary study and that a better analysis of the optimization problem and solution should be further investigated. We hope that the performance improvements reported will further motivate such analyses.

## Footnotes

[1]To clarify the difference between similar acronyms, a PDS function corresponds to a PSD matrix [4].

## References

[1] A. Argyriou, R. Hauser, C. Micchelli, and M. Pontil. A DC-programming algorithm for kernel selection. In *International Conference on Machine Learning*, 2006.

[2] A. Argyriou, C. Micchelli, and M. Pontil. Learning convex combinations of continuously parameterized basic kernels. In *Conference on Learning Theory*, 2005.

[3] F. Bach. Exploring large feature spaces with hierarchical multiple kernel learning. In *Advances in Neural Information Processing Systems*, 2008.

[4] C. Berg, J. P. R. Christensen, and P. Ressel. *Harmonic Analysis on Semigroups*. Springer-Verlag: Berlin-New York, 1984.

[5] J. Blitzer, M. Dredze, and F. Pereira. Biographies, Bollywood, Boom-boxes and Blenders: Domain Adaptation for Sentiment Classification. In *Association for Computational Linguistics*, 2007.

[6] B. Boser, I. Guyon, and V. Vapnik. A training algorithm for optimal margin classifiers. In *Conference on Learning Theory*, 1992.

[7] O. Chapelle, V. Vapnik, O. Bousquet, and S. Mukherjee. Choosing multiple parameters for support vector machines. *Machine Learning*, 46(1-3), 2002.

[8] C. Cortes, M. Mohri, and A. Rostamizadeh. Learning sequence kernels. In *Machine Learning for Signal Processing*, 2008.

[9] C. Cortes, M. Mohri, and A. Rostamizadeh. $L_2$ regularization for learning kernels. In *Uncertainty in Artificial Intelligence*, 2009.

[10] C. Cortes and V. Vapnik. Support-Vector Networks. *Machine Learning*, 20(3), 1995.

[11] T. Jebara. Multi-task feature and kernel selection for SVMs. In *International Conference on Machine Learning*, 2004.

[12] G. Lanckriet, N. Cristianini, P. Bartlett, L. E. Ghaoui, and M. Jordan. Learning the kernel matrix with semidefinite programming. *Journal of Machine Learning Research*, 5, 2004.

[13] C. Micchelli and M. Pontil. Learning the kernel function via regularization. *Journal of Machine Learning Research*, 6, 2005.

[14] C. S. Ong, A. Smola, and R. Williamson. Learning the kernel with hyperkernels. *Journal of Machine Learning Research*, 6, 2005.

[15] A. Rakotomamonjy, F. Bach, Y. Grandvalet, and S. Canu. Simplemkl. *Journal of Machine Learning Research*, 9, 2008.

[16] C. Saunders, A. Gammerman, and V. Vovk. Ridge Regression Learning Algorithm in Dual Variables. In *International Conference on Machine Learning*, 1998.

[17] B. Schölkopf and A. Smola. *Learning with Kernels*. MIT Press: Cambridge, MA, 2002.

[18] B. Scholkopf, A. Smola, and K. Muller. Nonlinear component analysis as a kernel eigenvalue problem. *Neural computation*, 10(5), 1998.

[19] J. Shawe-Taylor and N. Cristianini. *Kernel Methods for Pattern Analysis*. Cambridge University Press, 2004.

[20] S. Sonnenburg, G. Rätsch, C. Schäfer, and B. Schölkopf. Large scale multiple kernel learning. *Journal of Machine Learning Research*, 7, 2006.

[21] N. Srebro and S. Ben-David. Learning bounds for support vector machines with learned kernels. In *Conference on Learning Theory*, 2006.

[22] V. N. Vapnik. *Statistical Learning Theory*. Wiley-Interscience, New York, 1998.

[23] M. Varma and B. R. Babu. More generality in efficient multiple kernel learning. In *International Conference on Machine Learning*, 2009.

